# Kohonen Feature Maps and Growing Cell Structures – a Performance Comparison

**Bernd Fritzke**
International Computer Science Institute
1947 Center Street, Suite 600
Berkeley, CA 94704-1105, USA

## Abstract

A performance comparison of two self-organizing networks, the Kohonen Feature Map and the recently proposed Growing Cell Structures is made. For this purpose several performance criteria for self-organizing networks are proposed and motivated. The models are tested with three example problems of increasing difficulty. The Kohonen Feature Map demonstrates slightly superior results only for the simplest problem. For the other more difficult and also more realistic problems the Growing Cell Structures exhibit significantly better performance by every criterion. Additional advantages of the new model are that all parameters are constant over time and that size as well as structure of the network are determined automatically.

## 1 INTRODUCTION

Self-organizing networks are able to generate interesting low-dimensional representations of high-dimensional input data. The most well-known of these models is the Kohonen Feature Map (Kohonen [1982]). So far it has been applied to a large variety of problems including vector quantization (Schweizer et al. [1991]), biological modelling (Obermayer, Ritter & Schulten [1990]), combinatorial optimization (Favata & Walker [1991]) and also processing of symbolic information(Ritter & Kohonen [1989]).

It has been reported by a number of researchers, that one disadvantage of Kohonen's model is the fact, that the network structure had to be specified in advance. This is generally not possible in an optimal way since a necessary piece of information, the probability distribution of the input signals, is usually not available. The choice of an unsuitable network structure, however, can badly degrade network performance.

Recently we have proposed a new self-organizing network model – the Growing Cell Structures – which can automatically determine a problem specific network structure (Fritzke [1992]). By now the model has been successfully applied to clustering (Fritzke [1991]) and combinatorial optimization (Fritzke & Wilke [1991]).

In this contribution we directly compare our model to that of Kohonen. We first review some general properties of self-organizing networks and several performance criteria for these networks are proposed and motivated. The new model is then briefly described. Simulation results are presented and allow a comparison of both models with respect to the proposed criteria.

## 2    SELF-ORGANIZING NETWORKS

### 2.1    CHARACTERISTICS

A self-organizing network consists of a set of neurons arranged in some topological structure which induces *neighborhood relations* among the neurons. An *n*-dimensional *reference vector* is attached to every neuron. This vector determines the specific *n*-dimensional input signal to which the neuron is maximally sensitive.

By assigning to every input signal the neuron with the nearest reference vector (according to a suitable norm), a mapping is defined from the space of all possible input signals onto the neural structure. A given set of reference vectors thus divides the input vector space into regions with a common nearest reference vector. These regions are commonly denoted as *Voronoi regions* and the corresponding partition of the input vector space is denoted *Voronoi partition*.

Self-organizing networks learn (change internal parameters) in an unsupervised manner from a stream of input signals. These input signals obey a generally unknown probability distribution. For each input signal the neuron with the nearest reference vector is determined, the so-called "best matching unit" (bmu). The reference vectors of the bmu *and* of a number of its topological neighbors are moved towards the input signal. The adaptation of topological neighbors distinguishes self-organization ("winner take most") from competitive learning where only the bmu is adapted ("winner take all").

### 2.2    PERFORMANCE CRITERIA

One can identify three main criteria for self-organizing networks. The importance of each criterion may vary from application to application.

**Topology Preservation**. This denotes two properties of the mapping defined by the network. We call the mapping *topology-preserving* if

a) similar input vectors are mapped onto identical or closely neighboring neurons and

b) neighboring neurons have similar reference vectors.

Property a) ensures, that small changes of the input vector cause correspondingly small changes in the position of the bmu. The mapping is *robust* against distortions of the input, a very important property for applications dealing with real, noisy data. Property b) ensures robustness of the inverse mapping. The topology preservation is especially interesting when the dimension of the input vectors is higher than the network dimension. Then the mapping reduces the data dimension but usually preserves important similarity relations among the input data.

**Modelling of Probability Distribution**. A set of reference vectors is said to *model the probability distribution*, if the local density of reference vectors in the input vector space approaches the probability density of the input vector distribution.

This property is desirable for two reasons. First, we get an implicit model of the unknown probability distribution underlying the input signals. Second, the network becomes *fault-tolerant* against damage, since every neuron is only "responsible" for a small fraction of all input vectors. If neurons are destroyed for some reason the mapping ability of the network degrades only proportionally to the number of the destroyed neurons (soft fail). This is a very desirable property for technical (as well as natural) systems.

**Minimization of Quantization Error**. The *quantization error* for a given input signal is the distance between this signal and the reference vector of the bmu. We call a set of reference vectors *error minimizing* for a given probability distribution if the mean quantization error is minimized.

This property is important, if the original signals have to be reconstructed from the reference vectors which is a very common situation in vector quantization. The quantization error in this case limits the accuracy of the reconstruction.

One should note that the optimal distribution of reference vectors for error minimization is generally different from the optimal distribution for distribution modelling.

## 3   THE GROWING CELL STRUCTURES

The Growing Cell Structures are a self-organizing network an important feature of which is the ability to automatically find a problem specific network structure through a growth process.

Basic building blocks are $k$-dimensional *hypertetrahedrons*: lines for $k = 1$, triangles for $k = 2$, tetrahedrons for $k = 3$ etc. The vertices of the hypertetrahedrons are the neurons and the edges denote neighborhood relations.

By insertion and deletion of neurons the structure is modified. This is done during a self-organization process which is similar to that in Kohonen's model. Input signals cause adaptation of the bmu and its topological neighbors. In contrast to Kohonen's model all parameters are constant including the width of the neighborhood around

the bmu where adaptation takes place. Only *direct* neighbors and the bmu itself are being adapted.

## 3.1    INSERTION OF NEURONS

To determine the positions where new neurons should be inserted the concept of a *resource* is introduced. Every neuron has a local resource variable and new neurons are always inserted near the neuron with the highest resource value. New neurons get part of the resource of their neighbors so that in the long run the resource is distributed evenly among all neurons.

Every input signal causes an increase of the resource variable of the best matching unit. Choices for the resource examined so far are

- the summed quantization error caused by the neuron
- the number of input signals received by the neuron

Always after a constant number of adaptation steps (e.g. 100) a new neuron is inserted. For this purpose the neuron with the highest resource is determined and the edge connecting it to the neighbor with the most different reference vector is "split" by inserting the new neuron. Further edges are added to rebuild a structure consisting only of $k$-dimensional hypertetrahedrons.

The reference vector of the new neuron is interpolated from the reference vectors belonging to the ending points of the split edge. The resource variable of the new neuron is initialized by subtracting some resource from its neighbors, the amount of which is determined by the reduction of their Voronoi regions through the insertion.

## 3.2    DELETION OF NEURONS

By comparing the fraction of all input signals which a specific neuron has received and the volume of its Voronoi region one can derive a local *estimate of the probability density* of the input vectors.

Those neurons, whose reference vectors fall into regions of the input vector space with a very low probability density, are regarded as "superfluous" and are removed. The result are problem-specific network structures potentially consisting of several separate sub networks and accurately modelling a given probability distribution.

# 4    SIMULATION RESULTS

A number of tests have been performed to evaluate the performance of the new model. One series is described in the following.

Three methods have been compared.

- a) Kohonen Feature Maps (KFM)
- b) Growing Cell Structures with quantization error as resource (GCS-1)
- c) Growing Cell Structures with number of input signals as resource (GCS-2)

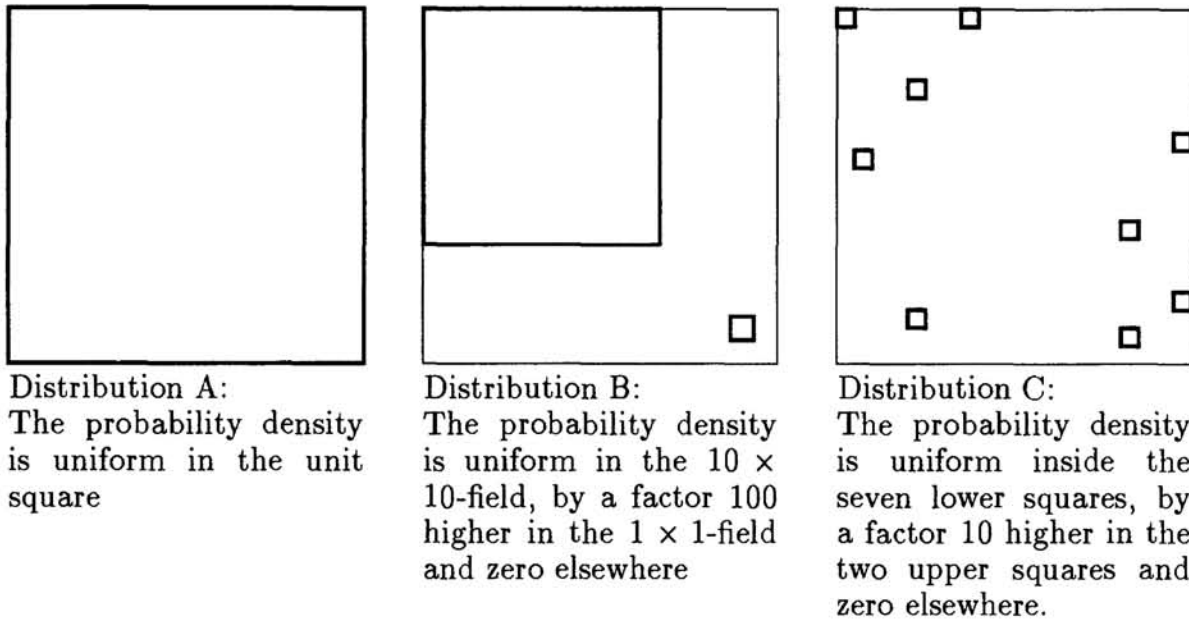

Distribution A:
The probability density is uniform in the unit square

Distribution B:
The probability density is uniform in the 10 × 10-field, by a factor 100 higher in the 1 × 1-field and zero elsewhere

Distribution C:
The probability density is uniform inside the seven lower squares, by a factor 10 higher in the two upper squares and zero elsewhere.

Figure 1: Three different probability distributions used for a performance comparison. Distribution A is very simple and has a form ideally suited for the Kohonen Feature Map which uses a square grid of neurons. Distribution B was chosen to show the effects of a highly varying probability density. Distribution C is the most realistic with a number of separate regions some of which have also different probability densities.

These models were applied to the probability distributions shown in fig. 1. The Kohonen model was used with a 10 × 10-grid of neurons. The Growing Cell Structures were used to build up a two dimensional cell structure of the same size. This was achieved by stopping the growth process when the number of neurons had reached 100.

At the end of the simulation the proposed criteria were measured as follows:

- The topology preservation requires two properties. Property a) was measured by the *topographical product* recently proposed by Bauer e.a. for this purpose (Bauer & Pawelzik [1992]). Property b) was measured by computing the *mean edge length* in the input space, i.e. the mean difference between reference vectors of directly neighboring neurons.

- The distribution modelling was measured by generating 5000 test signals according to the specific probability distribution and counting for every neuron the number of test signals it has been bmu for. The standard deviation of all counter values was computed and divided by the mean value of the counters to get a normalized measure, the *distribution error*, for the modelling of the probability distribution.

- The error minimization was measured by computing the *mean square quantization error* of the test signals.

The numerical results of the simulations are shown in fig. 2. Typical examples of the final network structures can be seen in fig. 3. It can be seen from fig. 2 that the

| model | A | B | C |
|-------|---|---|---|
| KFM | **0.0013** | 0.022 | 0.048 |
| GCS-1 | 0.0085 | 0.014 | 0.044 |
| GCS-2 | 0.0087 | **0.011** | **0.019** |

a) topographical product

| model | A | B | C |
|-------|---|---|---|
| KFM | **0.09** | 0.092 | 0.110 |
| GCS-1 | 0.11 | **0.056** | 0.015 |
| GCS-2 | 0.11 | 0.071 | **0.013** |

b) mean edge length

| model | A | B | C |
|-------|---|---|---|
| KFM | **0.20** | 0.84 | 0.90 |
| GCS-1 | 0.26 | **0.31** | **0.59** |
| GCS-2 | 0.26 | 1.57 | 0.73 |

c) distribution error

| model | A | B | C |
|-------|---|---|---|
| KFM | 0.0020 | 0.00077 | 0.00086 |
| GCS-1 | **0.0019** | 0.00089 | 0.00010 |
| GCS-2 | **0.0019** | **0.00055** | **0.00004** |

d) quantization error

Figure 2: Simulation results of the performance comparison. The model of Kohonen(KFM) and two versions of the Growing Cell Structures have been compared with respect to different criteria. All criteria are such, that smaller values are better values. The best (smallest) value in each column is enclosed in a box. Simulations were performed with the probability distributions A, B and C from fig. 1.

model of Kohonen has superior values only for distribution A, which is very regular and formed exactly like the chosen network structure (a square). Since generally the probability distribution is unknown and irregular, the distributions B and C are by far more realistic. For these distributions the Growing Cell Structures have the best values.

The modelling of the distribution and the minimization of the quantization error are generally concurring objectives. One has to decide which objective is more important for the current application. Then the appropriate version of the Growing Cell Structures can optimize with respect to that objective. For the complicated distribution C, however, *either* version of the Growing Cell Structures performs for every criterion better than Kohonen's model.

Especially notable is the low quantization error for distribution C and the error minimizing version (GCS-2) of the Growing Cell Structures (see fig. 2d ). This value indicates a good potential for vector quantization.

## 5   DISCUSSION

Our investigations indicate that – w.r.t the proposed criteria – the Growing Cell Structures are superior to Kohonen's model for all but very carefully chosen trivial examples. Although we used small examples for the sake of clarity, our experiments lead us to conjecture, that the difference will further increase with the difficulty and size of the problem.

There are some other important advantages of our approach. First, all parameters are constant. This eliminates the difficult choice of a "cooling schedule" which is necessary in Kohonen's model. Second, the network size does not have to be specified in advance. Instead the growth process can be continued until an arbitrary performance criterion is met. To meet a specific criterion with Kohonen's model, one generally has to try different network sizes. To start always with a very large

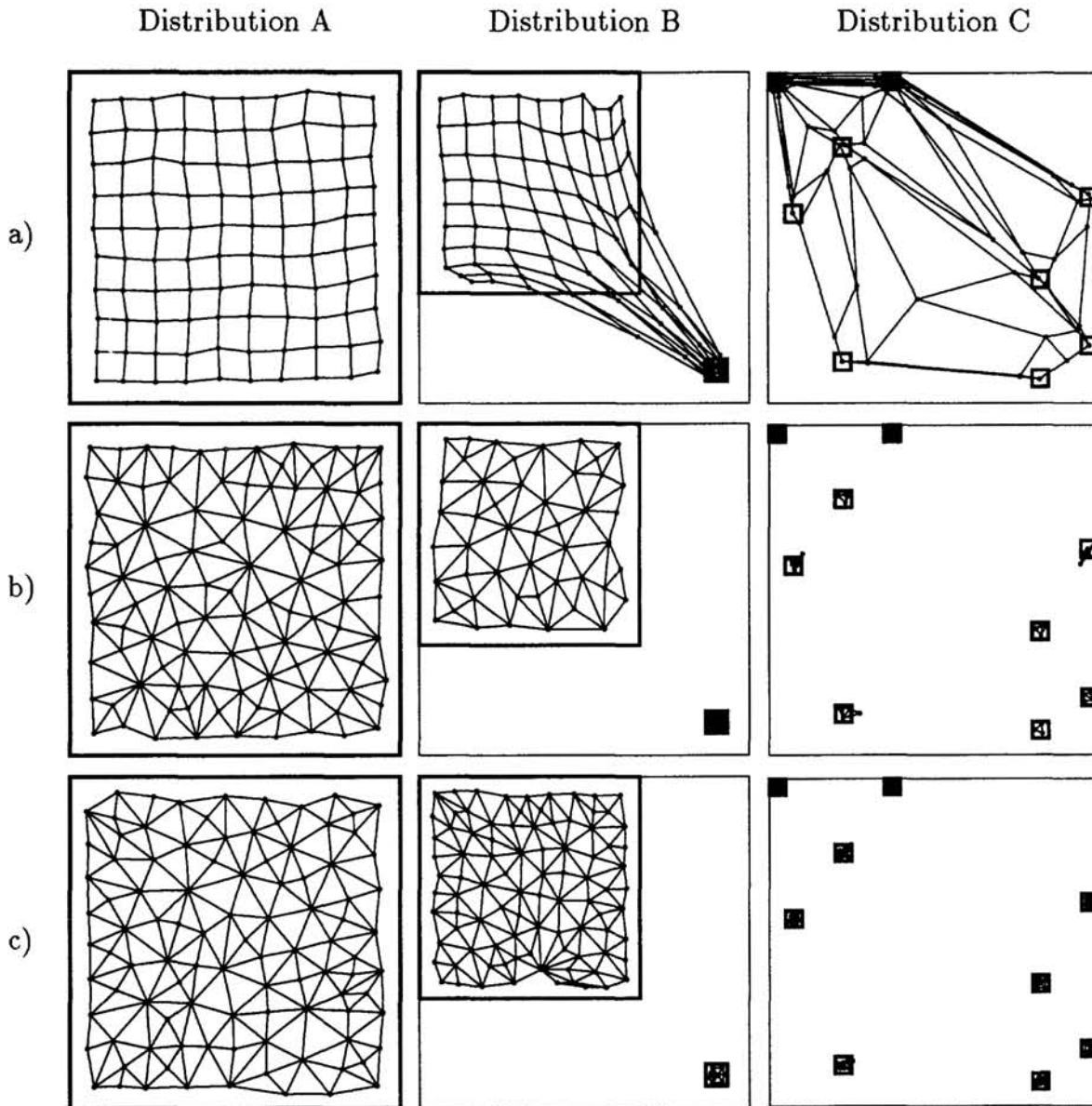

Figure 3: Typical simulation results for the model of Kohonen and the two versions of the Growing Cell Structures. The network size is 100 in every case. The probability distributions are described in fig. 1.

a) *Kohonen Feature Map* (KFM). For distributions B and C the fixed network structure leads to long connections and neurons in regions with zero probability density.

b) *Growing Cell Structures, distribution modelling variant* (GCS-1). The growth process combined with occasional removal of "superfluous" neurons has led to several sub networks for distributions B and C. For distribution B roughly half of the neurons are used to model either of the squares. This corresponds well to the underlying probability density.

c) *Growing Cell Structures, error minimizing variant* (GCS-2). The difference to the previous variant can be seen best for distribution B, where only a few neurons are used to cover the small square.

network is not a good solution to this problem, since the computational effort grows faster than quadratically with the network size.

Currently applications of variants of the new method to image compression and robot control are being investigated. Furthermore a new type of radial basis function network related to (Moody & Darken [1989]) is being explored, which is based on the Growing Cell Structures.

# REFERENCES

Bauer, H.-U. & K. Pawelzik [1992], "Quantifying the neighborhood preservation of self-organizing feature maps," *IEEE Transactions on Neural Networks* 3, 570–579.

Favata, F. & R. Walker [1991], "A study of the application of Kohonen-type neural networks to the travelling Salesman Problem," *Biological Cybernetics* 64, 463–468.

Fritzke, B. [1991], "Unsupervised clustering with growing cell structures," *Proc. of IJCNN-91*, Seattle, 531–536 (Vol. II).

Fritzke, B. [1992], "Growing cell structures – a self-organizing network in k dimensions," in *Artificial Neural Networks II*, I. Aleksander & J. Taylor, eds., North-Holland, Amsterdam, 1051–1056.

Fritzke, B. & P. Wilke [1991], "FLEXMAP - A neural network with linear time and space complexity for the traveling salesman problem," *Proc. of IJCNN-91*, Singapore, 929–934.

Kohonen, T. [1982], "Self-organized formation of topologically correct feature maps," *Biological Cybernetics* 43, 59–69.

Moody, J. & C. Darken [1989], "Fast Learning in Networks of Locally-Tuned Processing Units," *Neural Computation* 1, 281–294.

Obermayer, K., H. Ritter & K. Schulten [1990], "Large-scale simulations of self-organizing neural networks on parallel computers: application to biological modeling," *Parallel Computing* 14, 381–404.

Ritter, H.J. & T. Kohonen [1989], "Self-Organizing Semantic Maps," *Biological Cybernetics* 61, 241–254.

Schweizer, L., G. Parladori, G.L. Sicuranza & S. Marsi [1991], "A fully neural approach to image compression," in *Artificial Neural Networks*, T. Kohonen, K. Mäkisara, O. Simula & J. Kangas, eds., North-Holland, Amsterdam, 815–820.
